# Analog Neural Networks of Limited Precision I: Computing with Multilinear Threshold Functions (Preliminary Version)

**Zoran Obradovic and Ian Parberry**
Department of Computer Science,
Penn State University,
University Park, Pa. 16802.

## ABSTRACT

Experimental evidence has shown analog neural networks to be extremely fault-tolerant; in particular, their performance does not appear to be significantly impaired when precision is limited. Analog neurons with limited precision essentially compute k-ary weighted multilinear threshold functions, which divide $\mathbf{R}^n$ into $k$ regions with $k-1$ hyperplanes. The behaviour of *k-ary neural networks* is investigated. There is no canonical set of threshold values for $k>3$, although they exist for binary and ternary neural networks. The weights can be made integers of only $O((z+k)\log(z+k))$ bits, where z is the number of processors, without increasing hardware or running time. The weights can be made $\pm1$ while increasing running time by a constant multiple and hardware by a small polynomial in z and k. Binary neurons can be used if the running time is allowed to increase by a larger constant multiple and the hardware is allowed to increase by a slightly larger polynomial in z and k. Any symmetric k-ary function can be computed in constant depth and size $O(n^{k-1}/(k-2)!)$, and any k-ary function can be computed in constant depth and size $O(nk^n)$. The *alternating neural networks* of Olafsson and Abu-Mostafa, and the *quantized neural networks* of Fleisher are closely related to this model.

# 1  INTRODUCTION

Neural networks are typically circuits constructed from processing units which compute simple functions of the form $f(w_1,...,w_n):\mathbf{R}^n \to S$ where $S \subseteq \mathbf{R}$, $w_i \in \mathbf{R}$ for $1 \le i \le n$, and

$$f(w_1,...,w_n)(x_1, \ldots, x_n) = g\left(\sum_{i=1}^{n} w_i x_i\right)$$

for some *output function* $g:\mathbf{R} \to S$. There are two choices for the set $S$ which are currently popular in the literature. The first is the *discrete model*, with $S=\mathbf{B}$ (where $\mathbf{B}$ denotes the Boolean set $\{0,1\}$). In this case, $g$ is typically a *linear threshold function* $g(x)=1$ iff $x \ge 0$, and $f$ is called a *weighted linear threshold function*. The second is the *analog model*, with $S=[0,1]$ (where $[0,1]$ denotes $\{r \in \mathbf{R} | 0 \le r \le 1\}$). In this case, $g$ is typically a monotone increasing function, such as the *sigmoid function* $g(x)=(1+c^{-x})^{-1}$ for some constant $c \in \mathbf{R}$. The analog neural network model is popular because it is easy to construct processors with the required characteristics using a few transistors. The digital model is popular because its behaviour is easy to analyze.

Experimental evidence indicates that analog neural networks can produce accurate computations when the precision of their components is limited. Consider what actually happens to the analog model when the precision is limited. Suppose the neurons can take on k distinct excitation values (for example, by restricting the number of digits in their binary or decimal expansions). Then S is isomorphic to $\mathbf{Z}_k=\{0,...,k-1\}$. We will show that g is essentially the *multilinear threshold function* $g(h_1,h_2,...,h_{k-1}):\mathbf{R} \to \mathbf{Z}_k$ defined by

$$g(x)=i \;\; \textit{iff}\;\; h_i \le x < h_{i+1}.$$

Here and throughout this paper, we will assume that $h_1 \le h_2 \le ... \le h_{k-1}$, and for convenience define $h_0=-\infty$ and $h_k=\infty$. We will call $f$ a *k-ary weighted multilinear threshold function* when $g$ is a multilinear threshold function.

We will study neural networks constructed from k-ary multilinear threshold functions. We will call these *k-ary neural networks*, in order to distinguish them from the standard 2-ary or *binary neural network*. We are particularly concerned with the resources of *time*, *size* (number of processors), and *weight* (sum of all the weights) of k-ary neural networks when used in accordance with the classical computational paradigm. The reader is referred to (Parberry, 1990) for similar results on binary neural networks. A companion paper (Obradovic & Parberry, 1989b) deals with learning on k-ary neural networks. A more detailed version of this paper appears in (Obradovic & Parberry, 1989a).

# 2  A K-ARY NEURAL NETWORK MODEL

A *k-ary neural network* is a weighted graph $M=(V,E,w,h)$, where V is a set of processors and $E \subseteq V \times V$ is a set of connections between processors. Function $w:V \times V \to \mathbf{R}$ assign weights to interconnections and $h:V \to \mathbf{R}^{k-1}$ assign a set of $k-1$ thresholds to each of the processors. We assume that if $(u,v) \notin E$, $w(u,v)=0$. The *size* of M is defined to be the number of processors, and the *weight* of M is

$$\sum_{u,v\in V} |w(u,v)|.$$

The processors of a k-ary neural network are relatively limited in computing power. A *k-ary function* is a function $f:Z_k^n\to Z_k$. Let $F_k^n$ denote the set of all n-input k-ary functions. Define $\Theta_k^n:R^{n+k-1}\to F_k^n$ by $\Theta_k^n(w_1,...,w_n,h_1,...,h_{k-1}):R_k^n\to Z_k$, where

$$\Theta_k^n(w_1,...,w_n,h_1,...,h_{k-1})(x_1,...,x_n)=i \;\; iff \;\; h_i\le\sum_{i=1}^{n}w_i x_i <h_{i+1}.$$

The set of *k-ary weighted multilinear threshold functions* is the union, over all $n\in N$, of the range of $\Theta_k^n$. Each processor of a k-ary neural network can compute a k-ary weighted multilinear threshold function of its inputs.

Each processor can be in one of $k$ states, 0 through $k-1$. Initially, the input processors of M are placed into states which encode the input. If processor $v$ was updated during interval $t$, its state at time $t-1$ was $i$ and output was $j$, then at time $t$ its state will be $j$. A k-ary neural network *computes* by having the processors change state until a stable configuration is reached. The *output* of M are the states of the output processors after a stable state has been reached. A neural network $M_2$ is said to be $f(t)$-*equivalent* to $M_1$ iff for all inputs $x$, for every computation of $M_1$ on input $x$ which terminates in time $t$ there is a computation of $M_2$ on input $x$ which terminates in time $f(t)$ with the same output. A neural network $M_2$ is said to be *equivalent* to $M_1$ iff it is $t$-equivalent to it.

## 3   ANALOG NEURAL NETWORKS

Let $f$ be a function with range [0,1]. Any limited-precision device which purports to compute $f$ must actually compute some function with range the k rational values $R_k=\{i/k-1|i\in Z_k,0\le i<k\}$ (for some $k\in N$). This is sufficient for all practical purposes provided $k$ is large enough. Since $R_k$ is isomorphic to $Z_k$, we will formally define the limited precision variant of $f$ to be the function $f_k:X\to Z_k$ defined by $f_k(x)=round(f(x).(k-1))$, where $round:R\to N$ is the natural rounding function defined by $round(x)=n$ iff $n-0.5\le x<n+0.5$.

**Theorem 3.1** : Let $f(w_1,...,w_n):R^n\to[0,1]$ where $w_i\in R$ for $1\le i\le n$, be defined by

$$f(w_1,...,w_n)(x_1,\ldots,x_n)=g(\sum_{i=1}^{n}w_i x_i)$$

where $g:R\to[0,1]$ is monotone increasing and invertible. Then $f(w_1,...,w_n)_k:R^n\to Z_k$ is a k-ary weighted multilinear threshold function.

**Proof:** It is easy to verify that $f(w_1,...,w_n)_k=\Theta_k^n(w_1,...,w_n,h_1,...,h_{k-1})$, where $h_i=g^{-1}((2i-1)/2(k-1))$. $\square$

Thus we see that analog neural networks with limited precision are essentially k-ary neural networks.

## 4  CANONICAL THRESHOLDS

Binary neural networks have the advantage that all thresholds can be taken equal to zero (see, for example, Theorem 4.3.1 of Parberry, 1990). A similar result holds for ternary neural networks.

**Theorem 4.1 :** For every n-input ternary weighted multilinear threshold function there is an equivalent $(n+1)$-input ternary weighted multilinear threshold function with threshold values equal to zero and one.

**Proof:** Suppose $w=(w_1,...,w_n)\in R^n$, $h_1,h_2\in R$. Without loss of generality assume $h_1<h_2$. Define $\hat{w}=(\hat{w}_1,...,\hat{w}_{n+1})\in R^{n+1}$ by $\hat{w}_i=w_i/(h_2-h_1)$ for $1\leq i\leq n$, and $\hat{w}_{n+1}=-h_1/(h_2-h_1)$. It can be demonstrated by a simple case analysis that for all $x=(x_1,...,x_n)\in Z_k^n$,

$$\Theta_3^n(w,h_1,h_2)(x)=\Theta_3^{n+1}(\hat{w},0,1)(x_1,...,x_n,1).$$

□

The choice of threshold values in Theorem 4.1 was arbitrary. Unfortunately there is no canonical set of thresholds for $k>3$.

**Theorem 4.2 :** For every $k>3$, $n\geq2$, $m\geq0$, $h_1,...,h_{k-1}\in R$, there exists an n-input k-ary weighted multilinear threshold function

$$\Theta_k^n(w_1,...,w_n,t_1,...,t_{k-1}):Z_k^n\rightarrow Z_k,$$

such that for all $(n+m)$-input k-ary weighted multilinear threshold functions

$$\Theta_k^{n+m}(\hat{w}_1,....,\hat{w}_{n+m},h_1,...,h_{k-1}):Z_k^{m+n}\rightarrow Z_k$$

and $y_1,...,y_m\in R$, there exists $x=(x_1,...,x_n)\in Z_k^n$ such that

$$\Theta_k^n(w_1,...,w_n,t_1,...,t_{k-1})(x)\neq\Theta_k^{n+m}(\hat{w}_1,....,\hat{w}_{n+m},h_1,...,h_{k-1})(x_1,...x_n,y_1,...,y_m).$$

**Proof (Sketch):** Suppose that $t_1,...,t_{k-1}\in R$ is a canonical set of thresholds, and w.l.o.g. assume $n=2$. Let $h=(h_1,...,h_{k-1})$, where $h_1=h_2=2$, $h_3=4$, $h_i=5$ for $4\leq i<k$, and $f=\Theta_k^2(1,1,h)$.

By hypothesis there exist $w_1,...,w_{m+2}$ and $y=(y_1,...,y_m)\in R^m$ such that for all $x\in Z_k^2$,

$$f(x)=\Theta_k^{m+2}(w_1,...,w_{m+2},t_1,...,t_{k-1})(x,y).$$

Let $S=\sum_{i=1}^{m}w_{i+2}y_i$. Since $f(1,0)=0$, $f(0,1)=0$, $f(2,1)=2$, $f(1,2)=2$, it follows that

$$2(w_1+w_2+S)<t_1+t_3. \tag{1}$$

Since $f(2,0)=2$, $f(1,1)=2$, and $f(0,2)=2$, it follows that

$$w_1 + w_2 + S \geq t_2. \tag{2}$$

Inequalities (1) and (2) imply that

$$2t_2 < t_1 + t_3. \tag{3}$$

By similar arguments from $g = \Theta_k^2(1,1,1,3,3,4,...,4)$ we can conclude that

$$2t_2 > t_1 + t_3. \tag{4}$$

But (4) contradicts (3). $\square$

## 5  NETWORKS OF BOUNDED WEIGHT

Although our model allows each weight to take on an infinite number of possible values, there are only a finite number of threshold functions (since there are only a finite number of k-ary functions) with a fixed number of inputs. Thus the number of $n$-input threshold functions is bounded above by some function in $n$ and $k$. In fact, something stronger can be shown. All weights can be made integral, and $O((n+k)\log(n+k))$ bits are sufficient to describe each one.

**Theorem 5.1 :** For every k-ary neural network $M_1$ of size z there exists an equivalent k-ary neural network $M_2$ of size z and weight $((k-1)/2)^z(z+1)^{(z+k)/2+O(1)}$ with integer weights.

**Proof (Sketch):** It is sufficient to prove that for every weighted threshold function $f_k^n(w_1,...,w_n,h_1,...,h_{k-1}):Z_k^n \to Z_k$ for some $n \in N$, there is an equivalent weighted threshold function $g_k^n(w_1^*,...,w_n^*,h_1^*,...,h_{k-1}^*)$ such that $|w_i^*| \leq ((k-1)/2)^n(n+1)^{(n+k)/2+O(1)}$ for $1 \leq i \leq n$. By extending the techniques used by Muroga, Toda and Takasu (1961) in the binary case, we see that the weights are bounded above by the maximum determinant of a matrix of dimension $n+k-1$ over $Z_k$. $\square$

Thus if k is bounded above by a polynomial in n, we are guaranteed of being able to describe the weights using a polynomial number of bits.

## 6  THRESHOLD CIRCUITS

A k-ary neural network with weights drawn from $\{\pm 1\}$ is said to have *unit weights*. A unit-weight directed acyclic k-ary neural network is called a *k-ary threshold circuit*. A k-ary threshold circuit can be divided into layers, with each layer receiving inputs only from the layers above it. The *depth* of a k-ary threshold circuit is defined to be the number of layers. The weight is equal to the number of edges, which is bounded above by the square of the size. Despite the apparent handicap of limited weights, k-ary threshold circuits are surprisingly powerful.

Much interest has focussed on the computation of symmetric functions by neural networks, motivated by the fact that the visual system appears to be able to recognize objects regardless of their position on the retina. A function $f:Z_k^n \to Z_k$ is called *symmetric* if its output remains the same no matter how the input is permuted.

**Theorem 6.1** : Any symmetric k-ary function on n inputs can be computed by a k-ary threshold circuit of depth 6 and size $(n+1)^{k-1}/(k-2)! + O(kn)$.

**Proof:** Omitted. □

It has been noted many times that neural networks can compute any Boolean function in constant depth. The same is true of k-ary neural networks, although both results appear to require exponential size for many interesting functions.

**Theorem 6.2** : Any k-ary function of n inputs can be computed by a k-ary threshold circuit with size $(2n+1)k^n + k + 1$ and depth 4.

**Proof:** Similar to that for $k=2$ (see Chandra et. al., 1984; Parberry, 1990). □

The interesting problem remaining is to determine which functions require exponential size to achieve constant depth, and which can be computed in polynomial size and constant depth. We will now consider the problem of adding integers represented in k-ary notation.

**Theorem 6.3** : The sum of two k-ary integers of size $n$ can be computed by a k-ary threshold circuit with size $O(n^2)$ and depth 5.

**Proof:** First compute the carry of x and y in quadratic size and depth 3 using the standard elementary school algorithm. Then the $i^{th}$ position of the result can be computed from the $i^{th}$ position of the operands and a carry propagated in that position in constant size and depth 2. □

**Theorem 6.4** : The sum of n k-ary integers of size n can be computed by a k-ary threshold circuit with size $O(n^3 + kn^2)$ and constant depth.

**Proof:** Similar to the proof for $k=2$ using Theorem 6.3 (see Chandra et. al., 1984; Parberry, 1990). □

**Theorem 6.5** : For every k-ary neural network $M_1$ of size z there exists an $O(t)$-equivalent unit-weight k-ary neural network $M_2$ of size $O((z+k)^4 \log^3(z+k))$.

**Proof:** By Theorem 5.1 we can bound all weights to have size $O((z+k)\log(z+k))$ in binary notation. By Theorem 6.4 we can replace every processor with non-unit weights by a threshold circuit of size $O((z+k)^3 \log^3(z+k))$ and constant depth. □

Theorem 6.5 implies that we can assume unit weights by increasing the size by a polynomial and the running time by only a constant multiple provided the number of logic levels is bounded above by a polynomial in the size of the network. The number of thresholds can also be reduced to one if the size is increased by a larger polynomial:

**Theorem 6.6** : For every k-ary neural network $M_1$ of size z there exists an $O(t)$-equivalent unit-weight binary neural network $M_2$ of size $O(z^4 k^4)(\log z + \log k)^3$ which outputs the binary encoding of the required result.

**Proof:** Similar to the proof of Theorem 6.5. □

This result is primarily of theoretical interest. Binary neural networks appear simpler, and hence more desirable than analog neural networks. However, analog neural networks are actually more desirable since they are easier to build. With this in mind, Theorem 6.6 simply serves as a limit to the functions that an analog neural network

can be expected to compute efficiently. We are more concerned with constructing a model of the computational abilities of neural networks, rather than a model of their implementation details.

# 7   NONMONOTONE MULTILINEAR NEURAL NETWORKS

Olafsson and Abu-Mostafa (1988) study information capacity of functions $f(w_1,...,w_n):\mathbf{R}^n \to \mathbf{B}$ for $w_i \in \mathbf{R}$, $1 \leq i \leq n$, where

$$f(w_1,...,w_n)(x_1, \ldots, x_n)=g(\sum_{i=1}^{n} w_i x_i)$$

and g is the *alternating threshold function* $g(h_1,h_2,...,h_{k-1}):\mathbf{R} \to \mathbf{B}$ for some monotone increasing $h_i \in \mathbf{R}$, $1 \leq i < k$, defined by $g(x)=0$ if $h_{2i} \leq x < h_{2i+1}$ for some $0 \leq i \leq n/2$. We will call $f$ an *alternating weighted multilinear threshold function*, and a neural network constructed from functions of this form *alternating multilinear neural networks*. Alternating multilinear neural networks are closely related to k-ary neural networks:

**Theorem 7.1** : For every k-ary neural network of size $z$ and weight w there is an equivalent alternating multilinear neural network of size $z \log k$ and weight $(k-1)w \log (k-1)$ which produces the output of the former in binary notation.

**Proof (Sketch):** Each k-ary gate is replaced by log $k$ gates which together essentially perform a "binary search" to determine each bit of the k-ary gate. Weights which increase exponentially are used to provide the correct output value. □

**Theorem 7.2** : For every alternating multilinear neural network of size $z$ and weight w there is a 3t-equivalent k-ary neural network of size $4z$ and weight $w+4z$.

**Proof (Sketch):** Without loss of generality, assume $k$ is odd. Each alternating gate is replaced by a k-ary gate with identical weights and thresholds. The output of this gate goes with weight one to a k-ary gate with thresholds $1,3,5,...,k-1$ and with weight minus one to a k-ary gate with thresholds $-(k-1),...,-3,-1$. The output of these gates goes to a binary gate with threshold $k$. □

Both k-ary and alternating multilinear neural networks are a special case of *nonmonotone multilinear neural networks*, where $g:\mathbf{R} \to \mathbf{R}$ is the defined by $g(x)=c_i$ *iff* $h_i \leq x < h_{i+1}$, for some monotone increasing $h_i \in \mathbf{R}$, $1 \leq i < k$, and $c_0,...,c_{k-1} \in \mathbf{Z}_k$. Nonmonotone neural networks correspond to analog neural networks whose output function is not necessarily monotone nondecreasing. Many of the result of this paper, including Theorems 5.1, 6.5, and 6.6, also apply to nonmonotone neural networks. The size, weight and running time of many of the upper-bounds can also be improved by a small amount by using nonmonotone neural networks instead of k-ary ones. The details are left to the interested reader.

# 8   MULTILINEAR HOPFIELD NETWORKS

A multilinear version of the Hopfield network called the *quantized neural network* has been studied by Fleisher (1987). Using the terminology of (Parberry, 1990), a quantized neural network is a simple symmetric k-ary neural network (that is, its interconnection pattern is an undirected graph without self-loops) with the additional property that all processors have an identical set of thresholds. Although the latter assumption

is reasonable for binary neural networks (see, for example, Theorem 4.3.1 of Parberry, 1990), and ternary neural networks (Theorem 4.1), it is not necessarily so for k-ary neural networks with $k>3$ (Theorem 4.2). However, it is easy to extend Fleisher's main result to give the following:

**Theorem 8.1 :** Any productive sequential computation of a simple symmetric k-ary neural network will converge.

# 9  CONCLUSION

It has been shown that analog neural networks with limited precision are essentially k-ary neural networks. If k is limited to a polynomial, then polynomial size, constant depth k-ary neural networks are equivalent to polynomial size, constant depth binary neural networks. Nonetheless, the savings in time (at most a constant multiple) and hardware (at most a polynomial) arising from using k-ary neural networks rather than binary ones can be quite significant. We do not suggest that one should actually construct binary or k-ary neural networks. Analog neural networks can be constructed by exploiting the analog behaviour of transistors, rather than using extra hardware to inhibit it. Rather, we suggest that k-ary neural networks are a tool for reasoning about the behaviour of analog neural networks.

### Acknowledgements

The financial support of the Air Force Office of Scientific Research, Air Force Systems Command, USAF, under grant numbers AFOSR 87-0400 and AFOSR 89-0168 and NSF grant CCR-8801659 to Ian Parberry is gratefully acknowledged.

### References

Chandra A. K., Stockmeyer L. J. and Vishkin U., (1984) "Constant depth reducibility," *SIAM J. Comput.*, vol. 13, no. 2, pp. 423-439.

Fleisher M., (1987) "The Hopfield model with multi-level neurons," *Proc. IEEE Conference on Neural Information Processing Systems*, pp. 278-289, Denver, CO.

Muroga S., Toda I. and Takasu S., (1961) "Theory of majority decision elements," *J. Franklin Inst.*, vol. 271., pp. 376-418.

Obradovic Z. and Parberry I., (1989a) "Analog neural networks of limited precision I: Computing with multilinear threshold functions (preliminary version)," Technical Report CS-89-14, Dept. of Computer Science, Penn. State Univ.

Obradovic Z. and Parberry I., (1989b) "Analog neural networks of limited precision II: Learning with multilinear threshold functions (preliminary version)," Technical Report CS-89-15, Dept. of Computer Science, Penn. State Univ.

Olafsson S. and Abu-Mostafa Y. S., (1988) "The capacity of multilevel threshold functions," *IEEE Trans. Pattern Analysis and Machine Intelligence*, vol. 10, no. 2, pp. 277-281.

Parberry I., (To Appear in 1990) "A Primer on the Complexity Theory of Neural Networks," in *A Sourcebook of Formal Methods in Artificial Intelligence*, ed. R. Banerji, North-Holland.
